# Combining Estimators Using Non-Constant Weighting Functions

**Volker Tresp\*and Michiaki Taniguchi**
Siemens AG, Central Research
Otto-Hahn-Ring 6
81730 München, Germany

## Abstract

This paper discusses the linearly weighted combination of estimators in which the weighting functions are dependent on the input. We show that the weighting functions can be derived either by evaluating the input dependent variance of each estimator or by estimating how likely it is that a given estimator has seen data in the region of the input space close to the input pattern. The latter solution is closely related to the mixture of experts approach and we show how learning rules for the mixture of experts can be derived from the theory about learning with missing features. The presented approaches are modular since the weighting functions can easily be modified (no retraining) if more estimators are added. Furthermore, it is easy to incorporate estimators which were not derived from data such as expert systems or algorithms.

## 1 Introduction

Instead of modeling the global dependency between input $x \in \Re^D$ and output $y \in \Re$ using a single estimator, it is often very useful to decompose a complex mapping

into simpler mappings in the form[1]

$$\hat{y}(x) = \frac{1}{n(x)} \sum_{i=1}^{M} h_i(x) NN_i(x) = \sum_{i=1}^{M} g_i(x) NN_i(x) \qquad (1)$$

$$n(x) = \sum_{i=1}^{M} h_i(x) \quad h_i(x) >= 0 \quad g_i(x) = \frac{h_i(x)}{n(x)}.$$

The weighting functions $h_i(x)$ act as soft switches for the modules $NN_i(x)$. In the mixture of experts (Jacobs *et al.*, 1991) the decomposition is learned in an unsupervised manner driven by the training data and the main goal is a system which learns quickly. In other cases, the individual modules are trained individually and then combined using Equation 1. We can distinguish two motivations: first, in the work on averaging estimators (Perrone, 1993, Meir, 1994, Breiman, 1992) the modules are trained using identical data and the weighting functions are constant and, in the simplest case, all equal to one. The goal is to achieve improved estimates by averaging the errors of the individual modules. Second, a decomposition as described in Equation 1 might represent some "natural" decomposition of the problem leading to more efficient representation and training (Hampshire and Waibel, 1989). A good example is a decomposition into analysis and action. $h_i(x)$ might be the probability of disease $i$ given the symptoms $x$, the latter consisting of a few dozen variables. The amount of medication the patient should take given disease $i$ on the other hand — represented by the output of module $NN_i(x)$ — might only depend on a few inputs such as weight, gender and age.[2] Similarly, we might consider $h_i(x)$ as the IF-part of the rule, evaluating the weight of the rule given $x$, and as $NN_i(x)$ the conclusion or action which should be taken under rule $i$ (compare Tresp, Hollatz and Ahmad, 1993). Equation 1 might also be the basis for biological models considering for example the role of neural modulators in the brain. Nowlan and Sejnowsky (1994) recently presented a biologically motivated filter selection model for visual motion in which modules provide estimates of the direction and amount of motion and weighting functions select the most reliable module.

In this paper we describe novel ways of designing the weighting functions. Intuitively, the weighting functions should represent the competence or the certainty of a module, given the available information $x$. One possible measure is related to the number of training data that a module has seen in the neighborhood of $x$. Therefore, $\hat{P}(x|i)$, which is an estimate of the distribution of the input data which were used to train module $i$ is an obvious candidate as weighting function. Alternatively, the certainty a module assigns to its own prediction, represented by the inverse of the variance $1/var(NN_i(x))$ is a plausible candidate for a weighting function. Both approaches seem to be the flip-sides of the same coin, and indeed, we can show that both approaches are extremes of a unified approach.

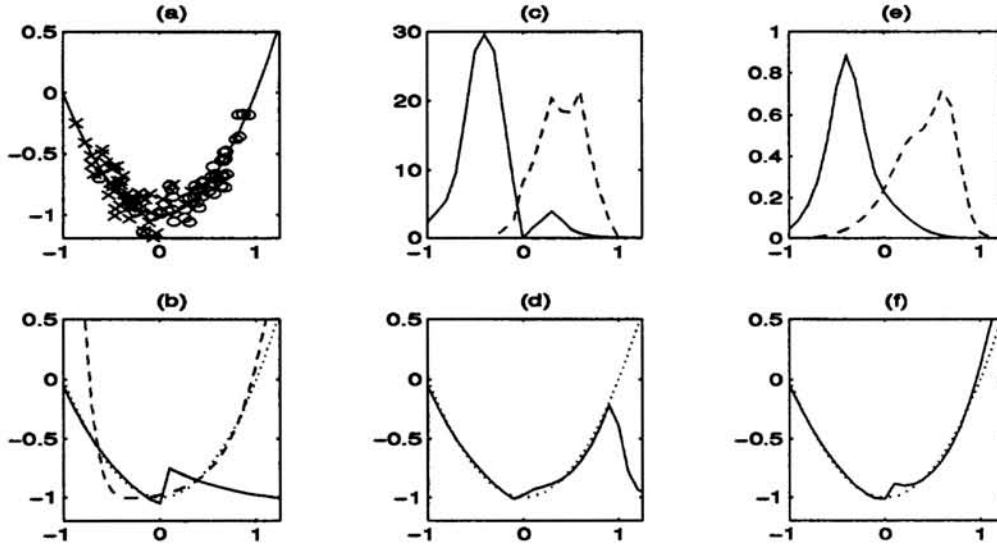

Figure 1: (a): Two data sets (1:*, 2:o ) and the underlying function (continuous). (b) The approximations of the two neural networks trained on the data sets (continuous: 1, dashed: 2). Note, that the approximation of a network is only reliable in the regions of the input space in which it has "seen" data. (c) The weighting functions for variance-based weighting. (d) The approximation using variance-based weighting (continuous). The approximation is excellent, except to the very right. (e) The weighting functions for density-based weighting (Gaussian mixtures approximation). (f) The approximation using density-based weighting (continuous). In particular to the right, the extrapolation is better than in (d).

## 2 Variance-based Weighting

Here, we assume that the different modules $NN_i(x)$ were trained with different data sets $\{(x_k^i, y_k^i)\}_{k=1}^{K_i}$ but that they model identical input-output relationships (see Figure 1 a,b ). To give a concrete example, this would correspond to the case that we trained two handwritten digit classifiers using different data sets and we want to use both for classifying new data.

If the errors of the individual modules are uncorrelated and unbiased,[3] the combined estimator is also unbiased and has the smallest variance if we select the weighting functions inversely proportional the the variance of the modules

$$h_i(x) = \frac{1}{var(NN_i(x))}. \tag{2}$$

This can be shown using $var(\sum_{i=1}^M g_i(x)NN_i(x)) = \sum_{i=1}^M g_i^2(x)var(NN_i(x))$ and using Lagrange multiplier to enforce the constraint that $\sum_i g_i(x) = 1$. Intuitively,

Equation 2 says that a module which is uncertain about its own prediction should also obtain a smaller weight. We estimate the variance from the training data as

$$var(NN_i(x)) \approx \frac{\partial NN_i(x)^T}{\partial w} H_i^{-1} \frac{\partial NN_i(x)}{\partial w}.$$

$H_i$ is the Hessian, which can be approximated as ($\sigma^2$ is the output-noise variance, Tibshirani, 1994)

$$H_i \approx \frac{1}{\sigma^2} \sum_{k=1}^{K_i} \frac{\partial NN_i(x_k^i)}{\partial w} \frac{\partial NN_i(x_k^i)}{\partial w}^T .$$

## 3   Density-based Weighting

In particular if the different modules were trained with data sets from different regions of the input space, it might be a reasonable assumption that the different modules represent different input-output relationships. In terms of our example, this corresponds to the problem, that we have two handwritten digit classifiers, one trained with American data and one with European data. If the classifiers are used in an international setting, confusions are possible, since, for example, an American seven might be confused with a European one. Formally, we introduce an additional variable which is equal to zero if the writer is American and is equal to one if the writer is European. During recall, we don't know the state of that variable and we are formally faced with the problem of estimation with missing inputs. From previous work (Ahmad and Tresp, 1993) we know that we have to integrate over the unknown input weighted by the conditional probability of the unknown input given the known variables. In this case, this translates into Equation 1, where the weighting function is

$$h_i(x) = P(i|x) = \frac{P(i,x)}{P(x)} \propto P(x|i)P(i).$$

In our example, $P(i|x)$ would estimate the probability that the writer is American or European given the data.

Depending on the problem $P(i|x)$ might be estimated in different ways. If $x$ represents continuous variables, we use a mixture of Gaussians model

$$\hat{P}(x|i) = \sum_j P^{ij} G(x; c^{ij}, \Sigma^{ij}) \quad \hat{P}(i) = \frac{K_i}{\sum_i K_i} \tag{3}$$

where $G(x; c^{ij}, \Sigma^{ij})$ is our notation for a normal density centered at $c^{ij}$ and with covariance $\Sigma^{ij}$.

Note that we have obtained a mixture of experts network with $\hat{P}(i|x)$ as gating network. A novel feature of our approach is that we maintain an estimate of the input data distribution (Equation 3), which is not modeled in the original mixture of experts network. This is advantageous if we have training data which are not assigned

to a module (in the mixture of experts, no data are assigned) which corresponds to training with missing inputs (the missing input is the missing assignment), for which the solution is known (Tresp *et al.*, 1994). If we use Gaussian mixtures to approximate $P(x|i)$, we can use generalized EM learning rules for adaptation. The adaptation of the parameters in the "gating network" which models $\hat{P}(x|i)$ is therefore somewhat simpler than in the original mixture of experts learning rules (see Section 8.2).

## 4   Unified Approach

In reality, the modules will often represent different mappings, but these mappings are not completely independent. Let's assume that we have an excellent American handwritten digit classifier but our European handwritten digit classifier is still very poor, since we only had few training data. We might want to take into account the results of the American classifier, even if we know that the writer was European. Mathematically, we can introduce a coupling between the modules. Let's assume that the prediction of the $i$-th module $NN_i(x) = f_i(x) + \epsilon_i$ is a noisy version of the true underlying relationship $f_i(x)$ and that $\epsilon_i$ is independent Gaussian noise with variance $var(NN_i(x))$. Furthermore, we assume that the true underlying functions are coupled through a prior distribution (for simplicity we only assume two modules)

$$P(f_1(x), f_2(x)) \propto \exp(-\frac{1}{2var_c}(f_1(x) - f_2(x))^2).$$

We obtain as best estimates

$$\hat{f}_1(x) = \frac{1}{K(x)}[(var(NN_2(x)) + var_c)\, NN_1(x) + var(NN_1(x))\, NN_2(x)]$$

$$\hat{f}_2(x) = \frac{1}{K(x)}[var(NN_2(x))\, NN_1(x) + (var(NN_1(x)) + var_c)\, NN_2(x)]$$

where

$$K(x) = var(NN_1(x)) + var(NN_2(x)) + var_c.$$

We use density-based weighting to combine the two estimates: $\hat{y}(x) = P(1|x)\hat{f}_1(x) + P(2|x)\hat{f}_2(x)$. Note, that if $var_c \to \infty$ (no coupling) we obtain the density-based solution and for $var_c \to 0$ (the mappings are forced to be identical) we obtain the variance-based solution. A generalization to more complex couplings can be found in Section 8.2.1.

## 5   Experiments

We tested our approaches using the Boston housing data set (13 inputs, one continuous output). The training data set consisted of 170 samples which were divided into 20 groups using k-means clustering. The clusters were then divided randomly into two groups and two multi-layer perceptrons (MLP) were trained using those two

data sets. Table 1 shows that the performances of the individual networks are pretty bad which indicates that both networks have only acquired local knowledge with only limited extrapolation capability. Variance-based weighting gives considerably better performance, although density-based weighting and the unified approach are both slightly better. Considering the assumptions, variance-based weighting should be superior since the underlying mappings are identical. One problem might be that we assumed that the modules are unbiased which might not be true in regions were a given module has seen no data.

Table 1: Generalization errors

| $NN_1$ | $NN_2$ | variance-based | density-based | unified |
|--------|--------|----------------|---------------|---------|
| 0.6948 | 1.188  | 0.4821         | 0.4472        | 0.4235  |

## 6  Error-based Weighting

In most learning tasks only one data set is given and the task is to obtain optimal predictions. Perrone (1994) has shown that simply averaging the estimates of a small number (i. e. 10) of neural network estimators trained on the same training data set often gives better performance than the best estimator out of this ensemble. Alternatively, bootstrap samples of the original data set can be used for training (Breimann, personal communication). Instead of averaging, we propose that Equation 1, where

$$h_i(x) = \frac{1}{var(NN_i(x)) + Res(NN_i(x))}$$

might give superior results (error-based weighting). $Res(NN_i(x))$ stands for an estimate of the input dependent residual squared error at $x$. As a simple approximation, $Res(NN_i(x))$ can be estimated by training a neural network with the residual squared errors of $NN_i$. Error-based weighting should be superior to simple averaging in particular if the estimators in the pool have different complexity. A more complex system would obtain larger weights in regions where the mapping is complex, since an estimator which is locally too simple has a large residual error, whereas in regions, where the mapping is simple, both estimators have sufficient complexity, but the simpler one has less variance. In our experiments we only tried networks with the same complexity. Preliminary results indicate that variance-based weighting and error-based weighting are sometimes superior to simple averaging. The main reason seems to be that the local overfitting of a network is reflected in a large variance near that location in input space. The overfitting estimator therefore obtains a small weight in that region (compare the overfitting of network 1 in Figure 1b near $x = 0$ and the small weight of network 1 close to $x = 0$ in Figure 1c).

# 7 Conclusions

We have presented modular ways for combining estimators. The weighting functions of each module can be determined independently of the other modules such that additional modules can be added without retraining of the previous system. This can be a useful feature in the context of the problem of *catastrophic forgetting*: additional data can be used to train an additional module and the knowledge in the remaining modules is preserved. Also note that estimators which are not derived from data can be easily included if it is possible to estimate the input dependent certainty or competence of that estimator.

**Acknowledgements:** Valuable discussions with David Cohn, Michael Duff and Cesare Alippi are greatfully acknowledged. The first author would like to thank the Center for Biological and Computational Learning (MIT) for providing and excellent research environment during the summer of 1994.

# 8 Appendix

## 8.1 Variance-based Weighting: Correlated Errors and Bias

We maintain that $\sum_i g_i(x) = 1$. In general (i.e. the modules have seen the same data, or partially the same data), we cannot assume that the errors in the individual modules are independent. Let the $M \times M$ matrix $\Omega(x)$ be the covariance between the predictions of the modules $NN_i(x)$. With $h(x) = (h_1(x)....h_M(x)^T$, the optimal weighting vector becomes

$$h(x) = \Omega^{-1}(x)\, u \quad n(x) = u'\, \Omega^{-1}(x)\, u$$

where $u$ is the $M$-dimensional unit vector.

If the individual modules are biased $(bias_i(x) = E_D(NN_i(x)) - E_{y|x}(y|x))$,[4] we form the $M \times M$ matrix $B(x)$, with $B_{ij}(x) = bias_i(x)bias_j(x)$, and the minimum variance solution is found for

$$h(x) = (\Omega(x) + B(x))^{-1}\, u \quad n(x) = u'\, (\Omega(x) + B(x))^{-1}\, u.$$

## 8.2 Density-based Weighting: GEM-learning

Let's assume a training pattern $(x_k, y_k)$ which is not associated with a particular module. If $w^i$ is a parameter in network $NN_i$ the error gradient becomes

$$\frac{\partial error_k}{\partial w^i} = -(y_k - NN_i(x_k))\, \hat{P}(i|x_k, y_k)\frac{\partial NN_i(x_k)}{\partial w^i}.$$

This equation can be derived from the solution to the problem of training with missing features (here: the true $i$ is unknown, see Tresp, Ahmad and Neuneier, 1994). This corresponds also to the M-step in a generalized EM algorithm, where the E-step calculates

$$\hat{P}(i|x_k, y_k) = \frac{\hat{P}(y_k|x_k, i)\hat{P}(x_k|i)\hat{P}(i)}{\sum_i \hat{P}(y_k|x_k, i)\hat{P}(x_k|i)\hat{P}(i)} \quad \hat{P}(y_k|x_k, i) = G(y_k; NN_i(x_k), \sigma^2).$$

using the current parameters. The M-step in the "gating network" $\hat{P}(x|i)$ is particularly simple using the well known EM-rules for Gaussian mixtures. Note, that $\hat{P}(\text{module} = i, \text{mixture component} : j|x_k, y_k)$ needs to be calculated.

### 8.2.1 Unified Approach: Correlated Errors and General Coupling

Let's form the vectors $NN(x) = (NN_1(x), ...NN_M(x))^T$ and $f(x) = (f_1(x), ..., f_M(x))^T$. In a more general case, the prior coupling between the underlying functions is described by

$$P(f(x)) = G(f(x); g(x), \Sigma_g(x))$$

where $g(x) = (g_1(x), ..., g_M(x))^T$. Furthermore, in a more general case, the estimates are not independent,

$$P(NN(x)|f(x)) = G(NN(x); f(x), \Sigma_N(x)).$$

The minimum variance solution is now

$$\hat{f}(x) = (\Sigma_N^{-1}(x) + \Sigma_g^{-1}(x))^{-1}(\Sigma_N^{-1}NN(x) + \Sigma_g^{-1}g(x)).$$

The equations in Section 4 are special cases with $M = 2$, $g(x) = 0$, $\Sigma_g^{-1}(x) = 1/var_c \times (1, -1)(1, -1)^T$, $\Sigma_N(x) = I (var(NN_1(x)), var(NN_2(x)))^T$ ($I$ is the 2D-unit matrix).

## Footnotes

\*At the time of the research for this paper, a visiting researcher at the Center for Biological and Computational Learning, MIT. Volker.Tresp@zfe.siemens.de

[1]The *hat* stands for an estimates value.

[2]Note, that we include the case that the weighting functions and the modules might explicitly only depend on different subsets of $x$.

[3]The errors are uncorrelated since the modules were trained with different data; correlation and bias are discussed in Section 8.1.

[4] $E$ stands for the expected value; the expectation $E_D$ is taken with respect to all data sets of the same size.

### References

Ahmad, S. and Tresp, V. (1993). Some Solutions to the Missing Feature Problem in Vision. In S. J. Hanson, J. D. Cowan and C. L. Giles, (Eds.), *Advances in Neural Information Processing Systems 5*. San Mateo, CA: Morgan Kaufmann.

Breiman, L. (1992). Stacked Regression. Dept. of Statistics, Berkeley, TR No. 367.

Hampshire, J. and Waibel, A. (1989). The meta-pi network: Building Distributed Knowledge Representations for Robust Pattern Recognition. TR CMU-CS-89-166, CMU, PA.

Jacobs, R. A., Jordan, M. I., Nowlan, S. J. and Hinton, J. E. (1991). Adaptive Mixtures of Local Experts. *Neural Computation*, Vol. 3, pp. 79-87.

Meir, R. (1994). Bias, Variance and the Combination of Estimators: The Case of Linear Least Squares. TR: Dept. of Electrical Engineering, Technion, Haifa.

Nowlan, S. J and Sejnowski, T. J. (1994). Filter Selection Model for Motion Segmentation and Velocity Integration. *J. Opt. Soc. Am. A*, Vol. 11, No. 12, pp. 1-24.

Perrone, M. P. (1993). *Improving Regression Estimates: Averaging Methods for Variance Reduction with Extensions to General Convex Measure Optimization*. PhD thesis. Brown University.

Tibshirani, R. (1994). A Comparison of Some Error Estimates for Neural Network Models. TR Department of Statistics, University of Toronto.

Tresp, V., Ahmad, S. and Neuneier, R. (1994). Training Neural Networks with Deficient Data. In: Cowan, J. D., Tesauro, G., and Alspector, J., eds., *Advances in Neural Information Processing Systems 6*, San Mateo, CA, Morgan Kaufman.

Tresp, V., Hollatz J. and Ahmad, S. (1993). Network Structuring and Training Using Rule-based Knowledge. In S. J. Hanson, J. D. Cowan and C. L. Giles, (Eds.), *Advances in Neural Information Processing Systems 5*, San Mateo, CA: Morgan Kaufmann.
